# Learning from Small Sample Sets by Combining Unsupervised Meta-Training with CNNs

**Yu-Xiong Wang**      **Martial Hebert**
Robotics Institute, Carnegie Mellon University
{yuxiongw, hebert}@cs.cmu.edu

## Abstract

This work explores CNNs for the recognition of novel categories from few examples. Inspired by the transferability properties of CNNs, we introduce an additional *unsupervised meta-training* stage that exposes multiple top layer units to a large amount of unlabeled real-world images. By encouraging these units to learn diverse sets of low-density separators across the unlabeled data, we capture a more generic, richer description of the visual world, which decouples these units from ties to a specific set of categories. We propose an unsupervised margin maximization that jointly estimates compact high-density regions and infers low-density separators. The *low-density separator (LDS)* modules can be plugged into any or all of the top layers of a standard CNN architecture. The resulting CNNs significantly improve the performance in scene classification, fine-grained recognition, and action recognition with small training samples.

## 1 Motivation

To successfully learn a deep convolutional neural network (CNN) model, hundreds of millions of parameters need to be inferred from millions of labeled examples on thousands of image categories [1, 2, 3]. In practice, however, for novel categories/tasks of interest, collecting a large corpus of annotated data to train CNNs from scratch is typically unrealistic, such as in robotics applications [4] and for customized categories [5]. Fortunately, although trained on particular categories, CNNs exhibit certain attractive transferability properties [6, 7]. This suggests that they could serve as universal feature extractors for novel categories, either as off-the-shelf features or through fine-tuning [7, 8, 9, 10].

Such transferability is promising but still restrictive, especially for novel-category recognition from few examples [11, 12, 13, 14, 15, 16, 17, 18]. The overall generality of CNNs is negatively affected by the *specialization of top layer units* to their original task. Recent analysis shows that from bottom, middle, to top layers of the network, features make a transition from general to specific [6, 8]. While features in the bottom and middle layers are fairly generic to many categories (i.e., low-level features of Gabor filters or color blobs and mid-level features of object parts), high-level features in the top layers eventually become specific and biased to best discriminate between a particular set of chosen categories. With limited samples from target tasks, fine-tuning cannot effectively adjust the units and would result in over-fitting, since it typically requires a significant amount of labeled data. Using off-the-shelf CNNs becomes the best strategy, despite the specialization and reduced performance.

In this work we investigate how to improve pre-trained CNNs for the learning from few examples. Our key insight is to expose multiple top layer units to *a massive set of unlabeled images*, as shown in Figure 1, which decouples these units from ties to the original specific set of categories. This additional stage is called *unsupervised meta-training* to distinguish this phase from the conventional unsupervised pre-training phase [19] and the training phase on the target tasks. Based on the above transferability analysis, intuitively, bottom and middle layers construct a feature space with *high-density regions* corresponding to potential latent categories. Top layer units in the pre-trained CNN, however, only have access to those regions associated with the original, observed categories. The

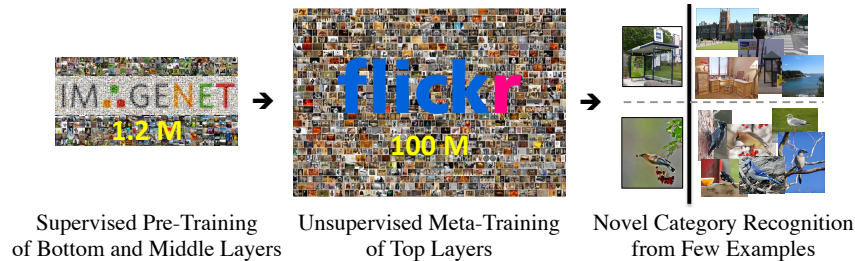

Supervised Pre-Training
of Bottom and Middle Layers

Unsupervised Meta-Training
of Top Layers

Novel Category Recognition
from Few Examples

Figure 1: We aim to improve the transferability of pre-trained CNNs for the recognition of novel categories from few labeled examples. We perform a multi-stage training procedure: 1) We first pre-train a CNN that recognizes a specific set of categories on a large-scale labeled dataset (e.g., ImageNet 1.2M), which provides fairly generic bottom and middle layer units; 2) We then meta-train the top layers as *low-density separators* on a far larger set of *unlabeled* data (e.g., Flickr 100M), which further improves the generality of multiple top layer units; 3) Finally, we use our modified CNN on new categories/tasks (e.g., scene classification, fine-grained recognition, and action recognition), either as off-the-shelf features or as initialization of fine-tuning that allows for end-to-end training.

units are then tuned to discriminate between these regions by separating the regions while pushing them further away from each other. To tackle this limitation, our unsupervised meta-training provides a far larger pool of unlabeled images as a much less biased sampling in the feature space. Now, instead of producing separations tied to the original categories, we generate diverse sets of separations across the unlabeled data. Since the unit "tries to discriminate the data manifold from its surroundings, in all non-manifold directions"[1], we capture a more generic and richer description of the visual world.

How can we generate these separations in an unsupervised manner? Inspired by the structure/manifold assumption in shallow semi-supervised and unsupervised learning (i.e., the decision boundary should not cross high-density regions, but instead lie in low-density regions) [20, 21], we introduce a *low-density separator* (LDS) module that can be plugged into any (or all) top layers of a standard CNN architecture. More precisely, the vector of weights connecting a unit to its previous layer (together with the non-linearity) can be viewed as a separator or decision boundary in the activation space of the previous layer. LDS then generates connection weights (decision boundaries) between successive layers that traverse regions of as low density as possible and avoid intersecting high-density regions in the activation space. Many LDS methods typically infer a probability distribution, for example through densest region detection, lowest-density hyperplane estimation [21], and clustering [22]. However, exact clustering or density estimation is known to be notoriously difficult in high-dimensional spaces.

We instead adopt a discriminative paradigm [20, 23, 24, 14] to circumvent the aforementioned difficulties. Using a max-margin framework, we propose an *unsupervised, scalable, coarse-to-fine* approach that jointly estimates compact, distinct high-density *quasi-classes* (HDQC), i.e., sets of data points sampled in high-density regions, as stand-ins for plausible high-density regions and infers low-density hyperplanes (separators). Our decoupled formulations generalize those in supervised binary code discovery [23] and semi-supervised learning [24], respectively; and more crucially, we propose a novel *combined* optimization to *jointly* estimate HDQC and learn LDS *in large-scale unsupervised* scenarios, from the labeled ImageNet 1.2M [25] to the unlabeled Flickr 100M dataset [26].

Our approach of exploiting unsupervised learning on top of CNN transfer learning is unique as opposed to other recent work on unsupervised, weakly-supervised, and semi-supervised deep learning. Most existing unsupervised deep learning approaches focus on unsupervised learning of visual representations that are both sparse and allow image reconstruction [19], including deep belief networks (DBN), convolutional sparse coding, and (denoising) auto-encoders (DAE). Our unsupervised LDS meta-training is different from conventional unsupervised pre-training as in DBN and DAE in two important ways: 1) our meta-training "post-arranges" the network that has undergone supervised training on a labeled dataset and then serves as a kind of network "pre-conditioner" [19] for the target tasks; and 2) our meta-training phase is not necessarily followed by fine-tuning and the features obtained by meta-training could be used off the shelf.

Other types of supervisory information (by creating auxiliary tasks), such as clustering, surrogate classes [27, 4], spatial context, temporal consistency, web supervision, and image captions [28], have been explored to train CNNs in an unsupervised (or weakly-supervised) manner. Although showing

initial promise, the performance of these unsupervised (or weakly-supervised) deep models is still not on par with that of their supervised counterparts, partially due to noisy or biased external information [28]. In addition, our LDS, if viewed as an auxiliary task, is directly related to discriminative classification, which results in more desirable and consistent features for the final novel-category recognition tasks. Unlike using a single image and its pre-defined transformations [27] or other labeled multi-view object [4] to simulate a surrogate class, our quasi-classes capture a more natural representation of realistic images. Finally, while we boost the overall generality of CNNs for a wide spectrum of unseen categories, semi-supervised deep learning approaches typically improve the model generalization for specific tasks, with both labeled and unlabeled data coming from the tasks of interest [29, 30].

**Our contribution** is three-fold: First, we show how LDS, based on an unsupervised margin maximization, is generated without a bias to a particular set of categories (Section 2). Second, we detail how to use LDS modules in CNNs by plugging them into any (or all) top layers of the architecture, leading to single-scale (or multi-scale) low-density separator networks (Section 3). Finally, we show how such modified CNNs, with enhanced generality, are used to facilitate the recognition of novel categories from few examples and significantly improve the performance in scene classification, fine-grained recognition, and action recognition (Section 4). The general setup is depicted in Figure 1.

## 2 Pre-trained low-density separators from unsupervised data

Given a CNN architecture pre-trained on a specific set of categories, such as the ImageNet (ILSVRC) 1,000 categories, we aim to improve the generality of one of its top layers, e.g., the $k$-th layer. We fix the structures and weights of the layers from 1 to $k-1$, and view the activation of layer $k-1$ as a feature space. A unit $s$ in layer $k$ is fully connected to all the units in layer $k-1$ via a vector of weights $\boldsymbol{w}^s$. Each $\boldsymbol{w}^s$ corresponds to a particular decision boundary (partition) of the feature space. Intuitively, all the $\boldsymbol{w}^s$'s then jointly further discriminate between these 1,000 categories, enforcing that the new activations in layer $k$ are more similar within classes and more dissimilar between classes.

To make $\boldsymbol{w}^s$'s and the associated units in layer $k$ unspecific to the ImageNet 1,000 categories, we use a large amount of *unlabeled* images at the unsupervised meta-training stage. The layers from 1 to $k-1$ remain unchanged, which means that we still tackle the same feature space. The new unlabeled images now constitute a less biased sampling of the feature space in layer $k-1$. We introduce a new $k$-th layer with more units and encourage their unbiased exploration of the feature space. More precisely, we enforce that the units learn many diverse decision boundaries $\boldsymbol{w}^s$'s that traverse different low-density regions while avoiding intersecting high-density regions of the unsupervised data (untied to the original ImageNet categories). The set of possible arrangements of such decision boundaries is rich, meaning that we can potentially generalize to a broad range of categories.

### 2.1 Approach overview

We denote column vectors and matrices with italic bold letters. For each unlabeled image $\mathcal{I}_i$, where $i \in \{1, 2, \ldots, N\}$, let $\boldsymbol{x}_i \in \mathbb{R}^D$ and $\boldsymbol{\phi}_i \in \mathbb{R}^S$ be the vectorized activations in layers $k-1$ and $k$, respectively. Let $\boldsymbol{W}$ be the weights between the two layers, where $\boldsymbol{w}^s$ is the weight vector associated with the unit $s$ in layer $k$. For notational simplicity, $\boldsymbol{x}_i$ already includes a constant 1 as the last element and $\boldsymbol{w}^s$ includes the bias term. We then have $\phi_i^s = f\left(\boldsymbol{w}^{s^T} \boldsymbol{x}_i\right)$, where $f(\cdot)$ is a non-linear function, such as sigmoid or ReLU. The resulting activation spaces of layers $k-1$ and $k$ are denoted as $\mathcal{X}$ and $\mathcal{F}$, respectively.

To learn $\boldsymbol{w}^s$'s as low-density separators, we are supposed to have certain high-density regions which $\boldsymbol{w}^s$'s separate. However, accurate estimation of high-density regions is difficult. We instead generate quasi-classes as stand-ins for plausible high-density regions. We want samples with the same quasi-labels to be similar in activation spaces (constraint within quasi-classes), while those with different quasi-labels should be very dissimilar in activation spaces (constraints between quasi-classes). Note that in contrast to clustering, generating quasi-classes does not require inferring membership for each data point. Formally, assuming that there are $C$ desired quasi-classes, we introduce a sample selection vector $\boldsymbol{T}_c \in \{0,1\}^N$ for each quasi-class $c$. $T_{c,i} = 1$ if $\mathcal{I}_i$ is selected for assignment to quasi-class $c$ and zero otherwise. As illustrated in Figure 4, the optimization for seeking low-density separators (LDS) while identifying high-density quasi-classes (HDQC) can be framed as

$$\begin{aligned} \text{find} \quad & \boldsymbol{W} \in \text{LDS}, \ \boldsymbol{T} \in \text{HDQC} \\ \text{subject to} \quad & \boldsymbol{W} \text{ separate } \boldsymbol{T}. \end{aligned} \quad (1)$$

This optimization problem enforces that each unit $s$ learns a partition $\boldsymbol{w}^s$ lying across the low-density region among certain salient high-density quasi-classes discovered by $\boldsymbol{T}$. This leads to a difficult joint optimization problem in theory, because $\boldsymbol{W}$ and $\boldsymbol{T}$ are interdependent.

In practice, however, it may be unnecessary to find the global optimum. Reasonable local optima are sufficient in our case to describe the feature space, as shown by the empirical results in Section 4. We use an iterative approach that obtains salient high-density quasi-classes from coarse to fine (Section 2.3) and produces promising discriminative low-density partitions among them (Section 2.2). We found that the optimization procedures converge in our experiments.

## 2.2 Learning low-density separators

Assume that $\boldsymbol{T}$ is known, which means that we have already defined $C$ high-density quasi-classes by $\boldsymbol{T}_c$. We then use a max-margin formulation to learn $\boldsymbol{W}$. Each unit $s$ in layer $k$ corresponds to a low-density hyperplane $\boldsymbol{w}^s$ that separates positive and negative examples in a max-margin fashion. To train $\boldsymbol{w}^s$, we need to generate label variables $\boldsymbol{l}^s \in \{-1, 1\}$ for each $\boldsymbol{w}^s$, which label the samples in the quasi-classes either as positive (1) or negative ($-1$) training examples. We can stack all the labels for learning $\boldsymbol{w}^s$'s to form $\boldsymbol{L} = [\boldsymbol{l}^1, \ldots, \boldsymbol{l}^S]$. Moreover, in the activation space $\mathcal{F}$ of layer $k$, which is induced by the activation space $\mathcal{X}$ of layer $k-1$ and $\boldsymbol{w}^s$, it would be beneficial to further push for large inter-quasi-class and small intra-quasi-class distances. We achieve such properties by optimizing

$$
\begin{aligned}
\min_{\boldsymbol{W},\boldsymbol{L},\boldsymbol{\Phi}} \sum_{s=1}^{S} \|\boldsymbol{w}^s\|^2 &+ \eta \sum_{i=1}^{N} \sum_{s=1}^{S} I_i \left[1 - l_i^s \left(\boldsymbol{w}^{s\,T} \boldsymbol{x_i}\right)\right]_+ \\
&+ \frac{\lambda_1}{2} \sum_{c=1}^{C} \sum_{\substack{u=1 \\ v=1}}^{N} T_{c,u} T_{c,v} d\left(\boldsymbol{\phi}_u, \boldsymbol{\phi}_v\right) - \frac{\lambda_2}{2} \sum_{c'=1}^{C} \sum_{\substack{c''=1 \\ c'' \neq c'}}^{C} \sum_{\substack{p=1 \\ q=1}}^{N} T_{c',p} T_{c'',q} d\left(\boldsymbol{\phi}_p, \boldsymbol{\phi}_q\right),
\end{aligned}
\tag{2}
$$

where $d$ is a distance metric (e.g., square of Euclidean distance) in the activation space $\mathcal{F}$ of layer $k$. $[x]_+ = max\,(0, x)$ represents the hinge loss. Here we introduce an additional indicator vector $\boldsymbol{I} \in \{0, 1\}^N$ for all the quasi-classes. $I_i = 0$ if $\mathcal{I}_i$ is not selected for assignment to any quasi-class (i.e., $\sum_{c=1}^{C} T_{c,i} = 0$) and one otherwise. Note that $\boldsymbol{I}$ is actually sparse, since only a portion of unlabeled samples are selected as quasi-classes and only their memberships are estimated in $\boldsymbol{T}$.

The new objective is much easier to optimize compared to Eqn. (1), as it only requires producing the low-density separators $\boldsymbol{w}^s$ from known quasi-classes given $\boldsymbol{T}_c$. We then derive an algorithm to optimize problem (2) using block coordinate descent. Specifically, problem (2) can be viewed as a generalization of predictable discriminative binary codes in [23]: 1) compared with the fully labeled case in [23], Eqn. (2) introduces additional quasi-class indicator variables to handle the unsupervised scenario; 2) Eqn. (2) extends the specific binary-valued hash functions in [23] to general real-valued non-linear activation functions in neural networks.

We adopt a similar iterative optimization strategy as in [23]. To achieve a good local minimum, our insight is that there should be diversity in $\boldsymbol{w}^s$'s and we thus initialize $\boldsymbol{w}^s$'s as the top-$S$ orthogonal directions of PCA on data points belonging to the quasi-classes. We found that this initialization yields promising results that work better than random initialization and do not contaminate the pre-trained CNNs. For fixed $\boldsymbol{W}$, we update $\boldsymbol{\Phi}$ using stochastic gradient descent to achieve improved separation in the activation space $\mathcal{F}$ of layer $k$. This optimization is efficient if using ReLU as non-linearity. We use $\boldsymbol{\Phi}$ to update $\boldsymbol{L}$. $l_i^s = 1$ if $\phi_i^s > 0$ and zero otherwise. Using $\boldsymbol{L}$ as training labels, we then train $S$ linear SVMs to update $\boldsymbol{W}$. We iterate this process a fixed number of times—$2 \sim 4$ in practice, and we thus obtain the low-density separator $\boldsymbol{w}^s$ for each unit and construct the activation space $\mathcal{F}$ of layer $k$.

## 2.3 Generating high-density quasi-classes

In the previous section, we assumed $\boldsymbol{T}$ known and learned low-density separators between high-density quasi-classes. Now we explain how to find these quasi-classes. Given the activation space $\mathcal{X}$ of layer $k-1$ and the activation space $\mathcal{F}$ of layer $k$ (linked by the low-density separators $\boldsymbol{W}$ as weights), we need to generate $C$ high-density quasi-classes from the unlabeled data selected by $\boldsymbol{T}_c$. We hope that the quasi-classes are distinct and compact in the activation spaces. That is, we want samples belonging to the same quasi-classes to be close to each other in the activation spaces, while samples from different quasi-classes should be far from each other in the activation spaces. To this end, we propose a coarse-to-fine procedure that combines the seeding heuristics of $K$-means++ [31] and a max-margin formulation [24] to gradually augment confident samples into the quasi-classes. We suppose that each quasi-class contains at least $\tau_0$ images and at most $\tau$ images. Learning $\boldsymbol{T}$ includes the following steps:

**Skeleton Generation**. We first choose a single seed point $T_{c,i_c} = 1$ for each quasi-class using the $K$-means++ heuristics in the activation space $\mathcal{X}$ of layer $k-1$. All the seed points are now spread out as the skeleton of the quasi-classes.

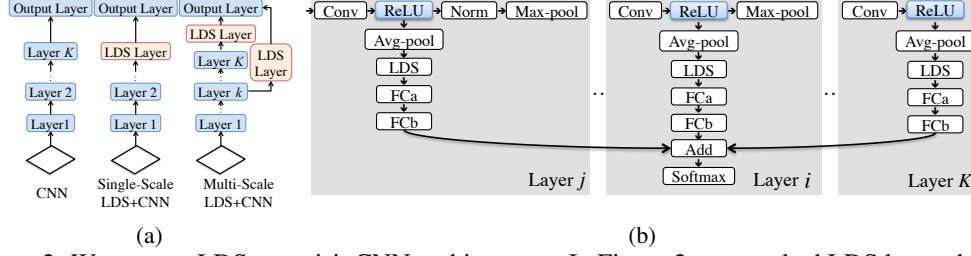

(a)                                           (b)

Figure 2: We use our LDS to revisit CNN architectures. In Figure 2a, we embed LDS learned from a large collection of unlabeled data as a new top layer into a standard CNN structure pre-trained on a specific set of categories (left), leading to single-scale LDS+CNN (middle). LDS could be also embedded into different layers, resulting multi-scale LDS+CNN (right). More specifically in Figure 2b, our multi-scale LDS+CNN architecture is constructed by introducing LDS layers into multi-scale DAG-CNN [10]. For each scale (level), we spatially (average) pool activations, learn and plug in LDS in this activation space, add fully-connected layers FCa and FCb (with $K$ outputs), and finally add the scores across all layers as predictions for $K$ output classes (that are finally soft-maxed together) on the target task. We show that the resulting LDS+CNNs can be either used as off-the-shelf features or discriminatively trained in an end-to-end fashion to facilitate novel category recognition.

**Quasi-Class Initialization**. We extend each single skeletal point to an initial quasi-class by adding its nearest neighbors [31] in the activation space $\mathcal{X}$ of layer $k-1$. Each of the resulting quasi-classes thus contains $\tau_0$ images, which satisfies the constraint for the minimum number of selected samples.

**Augmentation and Refinement**. In the above two steps, we select samples for quasi-classes based on the similarity in the activation space of layer $k-1$. Given this initial estimate of quasi-classes, we select additional samples using joint similarity in both activation spaces of layers $k-1$ and $k$ by leveraging a max-margin formulation. For each quasi-class $c$, we construct quasi-class classifiers $\boldsymbol{h}_c^{\mathcal{X}}$ and $\boldsymbol{h}_c^{\mathcal{F}}$ in the two activation spaces. Note that $\boldsymbol{h}_c^{\mathcal{X}}$ and $\boldsymbol{h}_c^{\mathcal{F}}$ are different from the low-density separator $\boldsymbol{w}^s$. We use SVM responses to select additional samples, leading to the following optimization:

$$
\min_{\boldsymbol{T},\boldsymbol{h}_c^{\mathcal{X}},\boldsymbol{h}_c^{\mathcal{F}}} \alpha \sum_{c=1}^{C} \left( \left\| \boldsymbol{h}_c^{\mathcal{X}} \right\|_2^2 + \lambda_{\mathcal{X}} \sum_{i=1}^{N} I_i \left[ 1 - y_{c,i} \left( \boldsymbol{h}_c^{\mathcal{X}^T} \boldsymbol{x}_i \right) \right]_+ \right) + \sum_{c'=1}^{C} \sum_{\substack{c''=1 \\ c' \neq c''}}^{C} \sum_{j=1}^{N} T_{c',j} T_{c'',j}
$$

$$
+ \beta \sum_{c=1}^{C} \left( \left\| \boldsymbol{h}_c^{\mathcal{F}} \right\|_2^2 + \lambda_{\mathcal{F}} \sum_{i=1}^{N} I_i \left[ 1 - y_{c,i} \left( \boldsymbol{h}_c^{\mathcal{F}^T} \boldsymbol{\phi}_i \right) \right]_+ - \sum_{j=1}^{N} T_{c,j} \left( \boldsymbol{h}_c^{\mathcal{F}^T} \boldsymbol{\phi}_j \right) \right)
$$

$$
s.t. \quad \tau_0 \leq \sum_{i=1}^{N} T_{c,i} \leq \tau, \forall c \in \{1,\ldots,C\}, \tag{3}
$$

where $y_{c,i}$ is the corresponding binary label used for one-vs.-all multi-quasi-class classification: $y_{c,i} = 1$ if $T_{c,i} = 1$ and $-1$ otherwise. The first and second terms denote a max-margin classifier in the activation space $\mathcal{X}$, and the fourth and fifth terms denote a max-margin classifier in the activation space $\mathcal{F}$. The third term ensures that the same unlabeled sample is not shared by multiple quasi-classes. The last term is a sample selection criterion that chooses those unlabeled samples with high classifier responses in the activation space $\mathcal{F}$.

This formulation is inspired by the approach to selecting unlabeled images using joint visual features and attributes [24]. We view our activation space $\mathcal{X}$ of layer $k-1$ as the feature space, and the activation space $\mathcal{F}$ of layer $k$ as the learned attribute space. However, different from the semi-supervised scenario in [24], which provides an initially labeled training images, our problem (3) is entirely unsupervised. To solve it, we use initial $\boldsymbol{T}$ corresponding to the quasi-classes obtained in the first two steps to train $\boldsymbol{h}_c^{\mathcal{X}}$ and $\boldsymbol{h}_c^{\mathcal{F}}$. After obtaining these two sets of SVMs in both activation spaces, we update $\boldsymbol{T}$. Following a similar block coordinate descent procedure as in [24], we iteratively re-train both $\boldsymbol{h}_c^{\mathcal{X}}$ and $\boldsymbol{h}_c^{\mathcal{F}}$ and update $\boldsymbol{T}$ until we obtain the desired $\tau$ number of samples.

## 3 Low-density separator networks

### 3.1 Single-scale layer-wise training

We start from how to embed our LDS as a new top layer into a standard CNN structure, leading to single-scale network. To improve the generality of the learned units in layer $k$, we need to prevent co-adaptation and enforce diversity between these units [6, 19]. We adopt a simple random sampling strategy to train the entire LDS layer. We break the units in layer $k$ into (disjoint) blocks, as shown

in Figure 4. We encourage each block of units to explore different regions of the activation space described by a random subset of unlabeled samples. This sampling strategy also makes LDS learning scalable since direct LDS learning from the entire dataset is computationally infeasible.

Specifically, from an original selection matrix $T_0 \in \{0, 1\}^{N \times C}$ of all zeros, we first obtain a random sub-matrix $T \in \{0, 1\}^{M \times C}$. Using this subset of $M$ samples, we then generate $C$ high-density quasi-classes by solving the problem (3) and learn $S$ corresponding low-density separator weights by solving the problem (2), yielding a block of $S$ units in layer $k$. We randomly produce $J$ sub-matrices $T$, repeat the procedure, and obtain $S \times J$ units ($J$ blocks) in total. This thus constitutes layer $k$, the low-density separator layer. The entire single-scale structure is shown in Figure 2a.

## 3.2 Multi-scale structure

For a convolutional layer of size $H_1 \times H_2 \times F$, where $H_1$ is the height, $H_2$ is the width, and $F$ is the number of filter channels, we first compute a $1 \times 1 \times F$ pooled feature by averaging across spatial dimensions as in [10], and then learn LDS in this activation space as before. Note that our approach applies to other types of pooling operation as well. Given the benefit of complementary features, LDS could also be operationalized on several different layers, leading to multi-scale/level representations. We thus modify the multi-scale DAG-CNN architecture [10] by introducing LDS on top of the ReLU layers, leading to multi-scale LDS+CNN, as shown in Figure 2b. We add two additional layers on top of LDS: FCa (with $F$ outputs) that selects discriminative units for target tasks, and FCb (with $K$ outputs) that learns $K$-way classifier for target tasks. The output of the LDS layers could be used as off-the-shelf multi-scale features. If using LDS weights as initialization, the entire structure in Figure 2b could also be fine-tuned in a similar fashion as DAG-CNN [10].

# 4 Experimental evaluation

In this section, we explore the use of low-density separator networks (LDS+CNNs) on a number of supervised learning tasks with limited data, including scene classification, fine-grained recognition, and action recognition. We use two powerful CNN models—AlexNet [1] and VGG19 [3] pre-trained on ILSVRC 2012 [25], as our reference networks. We implement the unsupervised meta-training on Yahoo! Flickr Creative Commons100M dataset (YFCC100M) [26], which is the largest single publicly available image and video database. We begin with plugging LDS into a single layer, and then introduce LDS into several top layers, leading to a multi-scale model. We consider using LDS+CNNs as off-the-shelf features in the small sample size regime, as well as through fine-tuning when enough data is available in the target task.

**Implementation Details.** During unsupervised meta-training, we use 99.2 million unlabeled images on YFCC100M [26]. After resizing the smallest side of each image to be 256, we generate the standard 10 crops (4 corners plus one center and their flips) of size $224 \times 224$ as implemented in Caffe [32]. For single-scale structures, we learn LDS in the $fc7$ activation space of dimension 4,096. For multi-scale structures, following [10] we learn LDS in activation spaces of $Conv3$, $Conv4$, $Conv5$, $fc6$, and $fc7$ for AlexNet, and we learn LDS in activation spaces of $Conv43$, $Conv44$, $Conv51$, $Conv52$, and $fc6$ for VGG19. We use the same sets of parameters to learn LDS in these activation spaces without further tuning. In the LDS layer, each block has $S = 10$ units, which separate across $M = 20,000$ randomly sub-sampled data points. Repeating $J = 2,000$ sub-sampling, we then have 20,000 units in total. Notably, each block of units in the LDS layer can be learned independently, making feasible for parallelization. For learning LDS in Eqn. (2), $\eta$ and $\lambda_1$ are set to 1 and $\lambda_2$ is set to normalize for the size of quasi-classes, which is the same setup and default parameters as in [23]. For generating high-density quasi-classes in Eqn. (3), following [31, 24], we set the minimum and maximum number of selected samples per quasi-classes to be $\tau_0 = 6$ and $\tau = 56$, and produce $C = 30$ quasi-classes in total. We use the same setup and parameters as in [24], where $\alpha = 1$, $\beta = 1$. While using only the center crops to infer quasi-classes, we use all 10 crops to learn more accurate LDS.

**Tasks and Datasets.** We evaluate on standard benchmark datasets for scene classification: SUN-397 [33] and MIT-67 [34], fine-grained recognition: Oxford 102 Flowers [35], and action recognition (compositional semantic recognition): Stanford-40 actions [36]. These datasets are widely used for evaluating the CNN transferability [8], and we consider their diversity and coverage of novel categories. We follow the standard experimental setup (e.g., the train/test splits) for these datasets.

## 4.1 Learning from few examples

The first question to answer is whether the LDS layers improve the transferability of the original pre-trained CNNs and facilitate the recognition of novel categories from few examples. To answer this

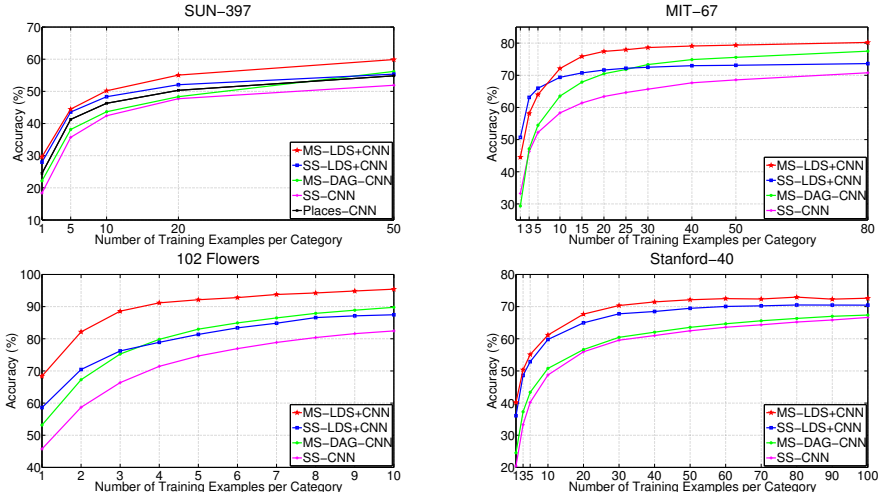

Figure 3: Performance comparisons between our single-scale LDS+CNN (SS-LDS+CNN), multi-scale LDS+CNN (MS-LDS+CNN) and the pre-trained single-scale CNN (SS-CNN), multi-scale DAG-CNN (MS-DAG-CNN) baselines for scene classification, fine-grained recognition, and action recognition from few labeled examples on four benchmark datasets. VGG19 [3] is used as the CNN model for its demonstrated superior performance. For SUN-397, we also include a publicly available strong baseline, Places-CNN, which trained a CNN (AlexNet architecture) from scratch using a scene-centric database with over 7 million annotated images from 400 scene categories, and which achieved state-of-the-art performance for scene classification [2]. X-axis: number of training examples per class. Y-axis: average multi-class classification accuracy. With improved transferability gained from a large set of unlabeled data, our LDS+CNNs with simple linear SVMs significantly outperform the vanilla pre-trained CNN and powerful DAG-CNN for small sample learning.

| Type | Approach | SUN-397 | MIT-67 | 102 Flowers | Stanford-40 |
|---|---|---|---|---|---|
| Weakly-supervised CNNs | Flickr-AlexNet | 42.7 | 55.8 | 74.2 | 53.0 |
| | Flickr-GoogLeNet | 44.4 | 55.6 | 65.8 | 52.8 |
| | Combined-AlexNet | 47.3 | 58.8 | 83.3 | 56.4 |
| | Combined-GoogLeNet | 55.0 | 67.9 | 83.7 | 69.2 |
| Ours | SS-LDS+CNN | 55.4 | 73.6 | 87.5 | 70.5 |
| | MS-LDS+CNN | **59.9** | **80.2** | **95.4** | **72.6** |

Table 1: Performance comparisons of classification accuracy (%) between our LDS+CNNs and weakly-supervised CNNs [28] on the four datasets when using the entire training sets. In contrast to our approach that uses the Flickr dataset for unsupervised meta-training, Flickr-AlexNet/GoogLeNet train CNNs from scratch on the Flickr dataset by using associated captions as weak supervisory information. Combined-AlexNet/GoogLeNet concatenate features from supervised ImageNet CNNs and weakly-supervised Flickr CNNs. Despite the same amount of data used for pre-training, ours outperform the weakly-supervised CNNs by a significant margin due to their noisy captions and tags.

question, we evaluate both LDS+CNN and CNN as off-the-shelf features without fine-tuning on the target datasets. This is the standard way to use pre-trained CNNs [7]. We test how performance varies with the number of training samples per category as in [16]. To compare with the state-of-the-art performance, we use VGG19 in this set of experiments. Following the standard practice, we train simple linear SVMs in one-vs.-all fashion on $L2$-normalized features [7, 10] in Liblinear [37].

**Single-Scale Features.** We begin by evaluating single-scale features on theses datasets. For a fair comparison, we first reduce the dimensionality of LDS+CNN from $20,000$ to $4,096$, the same dimensionality as CNN, followed by linear SVMs. This is achieved by selecting from LDS+CNN the $4,096$ most active features according to the standard criterion of multi-class recursive feature elimination (RFE) [38] using the target dataset. We also tested PCA. The performance drops, but it is still significantly better than the pre-trained CNN. Figure 3 summarizes the average performance over 10 random splits on these datasets. When used as off-the-shelf features for small-sample learning, our single-scale LDS+CNN significantly outperforms the vanilla pre-trained CNN, which is already a strong baseline. Our results are particularly impressive for the big performance boost, for example nearly $20\%$ on MIT-67, in the one-shot learning scenario. This verifies the effectiveness of the layer-wise LDS, which leads to a more generic representation for a broad range of novel categories.

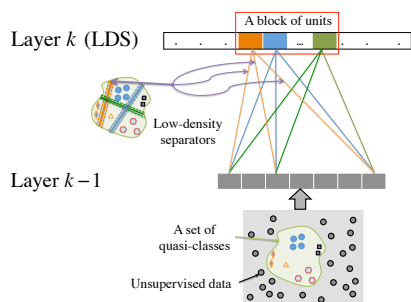

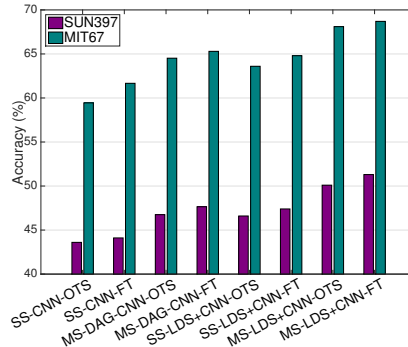

Figure 4: Illustration of learning low-density separators between successive layers on a large amount of unlabeled data. Note the color correspondence between the decision boundaries across the unlabeled data and the connection weights in the network.

Figure 5: Effect of fine-tuning (FT) on SUN-397 (purple bars) and MIT-67 (blue bars). Fine-tuning LDS+CNNs (AlexNet) further improves the performance over the off-the-shelf (OTS) features for novel category recognition.

**Multi-Scale Features.** Given the promise of single-scale LDS+CNN, we now evaluate multi-scale off-the-shelf features. After learning LDS in each activation space separately, we reduce their dimensionality to that of the corresponding activation space via RFE for a fair comparison with DAG-CNN [10]. We train linear SVMs on these LDS+CNNs, and then average their predictions. Figure 3 summarizes the average performance over different splits for multi-scale features. Consistent with the single-scale results, our multi-scale LDS+CNN outperforms the powerful multi-scale DAG-CNN. LDS+CNN is especially beneficial to fine-grained recognition, since there is typically limited data per class for fine-grained categories. Figure 3 also validates that multi-scale LDS+CNN allows for transfer at different levels, thus leading to better generalization to novel recognition tasks compared to its single-scale counterpart. In addition, Table 1 further shows that our LDS+CNNs outperform weakly-supervised CNNs [28] that are directly trained on Flickr using external caption information.

## 4.2 Fine-tuning

With more training data available in the target task, our LDS+CNNs could be fine-tuned to further improve the performance. For efficient and easy fine-tuning, we use AlexNet in this set of experiments as in [10]. We evaluate the effect of fine-tuning of our single-scale and multi-scale LDS+CNNs in the scene classification tasks, due to their relatively large number of training samples. We compare against the fine-tuned single-scale CNN and multi-scale DAG-CNN [10], as shown in Figure 5. For completeness, we also include their off-the-shelf performance. As expected, fine-tuned models consistently outperform their off-the-shelf counterparts. Importantly, Figure 5 shows that our approach is not limited to small-sample learning and is still effective *even in the many training examples regime*.

## 5 Conclusions

Even though current large-scale annotated datasets are comprehensive, they are only a tiny sampling of the full visual world biased to a selection of categories. It is still not clear how to take advantage of truly large sets of unlabeled real-world images, which constitute a much less biased sampling of the visual world. In this work we proposed an approach to leveraging such unsupervised data sources to improve the overall transferability of supervised CNNs and thus to facilitate the recognition of novel categories from few examples. This is achieved by encouraging multiple top layer units to generate diverse sets of low-density separations across the unlabeled data in activation spaces, which decouples these units from ties to a specific set of categories. The resulting modified CNNs (single-scale and multi-scale low-density separator networks) are fairly generic to a wide spectrum of novel categories, leading to significant improvement for scene classification, fine-grained recognition, and action recognition. The specific implementation described here is a first step. While we used certain max-margin optimization to train low-density separators, it would be interesting to integrate into the current CNN backpropagation framework both learning low-density separators and gradually estimating high-density quasi-classes.

**Acknowledgments**. We thank Liangyan Gui, Carl Doersch, and Deva Ramanan for valuable and insightful discussions. This work was supported in part by ONR MURI N000141612007 and U.S. Army Research Laboratory (ARL) under the Collaborative Technology Alliance Program, Cooperative Agreement W911NF-10-2-0016. We also thank NVIDIA for donating GPUs and AWS Cloud Credits for Research program.

## Footnotes

[1]Yoshua Bengio. `https://disqus.com/by/yoshuabengio/`

# References

[1] A. Krizhevsky, I. Sutskever, and G. E. Hinton. ImageNet classification with deep convolutional neural networks. In *NIPS*, 2012.

[2] B. Zhou, A. Lapedriza, J. Xiao, A. Torralba, and A. Oliva. Learning deep features for scene recognition using places database. In *NIPS*, 2014.

[3] K. Simonyan and A. Zisserman. Very deep convolutional networks for large-scale image recognition. In *ICLR*, 2015.

[4] D. Held, S. Thrun, and S. Savarese. Robust single-view instance recognition. In *ICRA*, 2016.

[5] Y.-X. Wang and M. Hebert. Model recommendation: Generating object detectors from few samples. In *CVPR*, 2015.

[6] J. Yosinski, J. Clune, Y. Bengio, and H. Lipson. How transferable are features in deep neural networks? In *NIPS*, 2014.

[7] A. S. Razavian, H. Azizpour, J. Sullivan, and S. Carlsson. CNN features off-the-shelf: An astounding baseline for recognition. In *CVPR Workshop*, 2014.

[8] H. Azizpour, A. S. Razavian, J. Sullivan, A. Maki, and S. Carlsson. Factors of transferability for a generic ConvNet representation. *TPAMI*, 2015.

[9] M. Oquab, L. Bottou, I. Laptev, and J. Sivic. Learning and transferring mid-level image representations using convolutional neural networks. In *CVPR*, 2014.

[10] S. Yang and D. Ramanan. Multi-scale recognition with DAG-CNNs. In *ICCV*, 2015.

[11] G. Koch, R. Zemel, and R. Salakhutdinov. Siamese neural networks for one-shot image recognition. In *ICML Workshops*, 2015.

[12] B. M. Lake, R. Salakhutdinov, and J. B. Tenenbaum. Human-level concept learning through probabilistic program induction. *Science*, 350(6266):1332–1338, 2015.

[13] O. Vinyals, C. Blundell, T. Lillicrap, K. Kavukcuoglu, and D. Wierstra. Matching networks for one shot learning. In *NIPS*, 2016.

[14] Y.-X. Wang and M. Hebert. Learning by transferring from unsupervised universal sources. In *AAAI*, 2016.

[15] Z. Li and D. Hoiem. Learning without forgetting. In *ECCV*, 2016.

[16] Y.-X. Wang and M. Hebert. Learning to learn: Model regression networks for easy small sample learning. In *ECCV*, 2016.

[17] L. Bertinetto, J. F. Henriques, J. Valmadre, P. Torr, and A. Vedaldi. Learning feed-forward one-shot learners. In *NIPS*, 2016.

[18] B. Hariharan and R. Girshick. Low-shot visual object recognition. arXiv preprint arXiv:1606.02819, 2016.

[19] I. Goodfellow, Y. Bengio, and A. Courville. Deep learning. Book in preparation for MIT Press, 2016.

[20] O. Chapelle and A. Zien. Semi-supervised classification by low density separation. In *AISTATS*, 2005.

[21] S. Ben-david, T. Lu, D. Pál, and M. Sotáková. Learning low density separators. In *AISTATS*, 2009.

[22] J. Hoffman, B. Kulis, T. Darrell, and K. Saenko. Discovering latent domains for multisource domain adaptation. In *ECCV*, 2012.

[23] M. Rastegari, A. Farhadi, and D. Forsyth. Attribute discovery via predictable discriminative binary codes. In *ECCV*, 2012.

[24] J. Choi, M. Rastegari, A. Farhadi, and L. S. Davis. Adding unlabeled samples to categories by learned attributes. In *CVPR*, 2013.

[25] O. Russakovsky, J. Deng, H. Su, J. Krause, S. Satheesh, S. Ma, Z. Huang, A. Karpathy, A. Khosla, M. Bernstein, A. C. Berg, and L. Fei-Fei. ImageNet large scale visual recognition challenge. *IJCV*, 115(3):211–252, 2015.

[26] B. Thomee, D. A. Shamma, G. Friedland, B. Elizalde, K. Ni, D. Poland, D. Borth, and L.-J. Li. YFCC100M: The new data in multimedia research. *Communications of the ACM*, 59(2):64–73, 2016.

[27] A. Dosovitskiy, J. T. Springenberg, M. Riedmiller, and T. Brox. Discriminative unsupervised feature learning with convolutional neural networks. In *NIPS*, 2014.

[28] A. Joulin, L. van der Maaten, A. Jabri, and N. Vasilache. Learning visual features from large weakly supervised data. In *ECCV*, 2016.

[29] J. Weston, F. Ratle, H. Mobahi, and R. Collobert. Deep learning via semi-supervised embedding. In *ICML*, 2008.

[30] A. Ahmed, K. Yu, W. Xu, Y. Gong, and E. P. Xing. Training hierarchical feed-forward visual recognition models using transfer learning from pseudo-tasks. In *ECCV*, 2008.

[31] D. Dai and L. Van Gool. Ensemble projection for semi-supervised image classification. In *ICCV*, 2013.

[32] Y. Jia, E. Shelhamer, J. Donahue, S. Karayev, J. Long, R. Girshick, S. Guadarrama, and T. Darrell. Caffe: Convolutional architecture for fast feature embedding. In *ACM MM*, 2014.

[33] J. Xiao, K. A. Ehinger, J. Hays, A. Torralba, and A. Oliva. SUN database: Exploring a large collection of scene categories. *IJCV*, 119(1):3–22, 2016.

[34] A. Torralba and A. Quattoni. Recognizing indoor scenes. In *CVPR*, 2009.

[35] M.-E. Nilsback and A. Zisserman. Automated flower classification over a large number of classes. In *ICVGIP*, 2008.

[36] B. Yao, X. Jiang, A. Khosla, A. L. Lin, L. Guibas, and L. Fei-Fei. Human action recognition by learning bases of action attributes and parts. In *ICCV*, 2011.

[37] R.-E. Fan, K.-W. Chang, C.-J. Hsieh, X.-R. Wang, and C.-J. Lin. LIBLINEAR: A library for large linear classification. *JMLR*, 9:1871–1874, 2008.

[38] A. Bergamo and L. Torresani. Classemes and other classifier-based features for efficient object categorization. *TPAMI*, 36(10):1988–2001, 2014.

